# Neural Dynamics of Motion Segmentation and Grouping

**Ennio Mingolla**
Center for Adaptive Systems, and
Cognitive and Neural Systems Program
Boston University
111 Cummington Street
Boston, MA 02215

## Abstract

A neural network model of motion segmentation by visual cortex is described. The model clarifies how preprocessing of motion signals by a Motion Oriented Contrast Filter (MOC Filter) is joined to long-range cooperative motion mechanisms in a motion Cooperative Competitive Loop (CC Loop) to control phenomena such as as induced motion, motion capture, and motion aftereffects. The total model system is a motion Boundary Contour System (BCS) that is computed in parallel with a static BCS before both systems cooperate to generate a boundary representation for three dimensional visual form perception. The present investigations clarify how the static BCS can be modified for use in motion segmentation problems, notably for analyzing how ambiguous local movements (the aperture problem) on a complex moving shape are suppressed and actively reorganized into a coherent global motion signal.

## 1    INTRODUCTION: WHY ARE STATIC AND MOTION BOUNDARY CONTOUR SYSTEMS NEEDED?

Some regions, notably MT, of visual cortex are specialized for motion processing. However, even the earliest stages of visual cortex processing, such as simple cells in V1, require stimuli that change through time for their maximal activation and are direction-sensitive. Why has evolution generated regions such as MT, when even V1 is change-sensitive and direction-sensitive? What computational properties are achieved by MT that are not already available in V1?

The monocular Boundary Contour System (BCS) theory of Grossberg and Mingolla (1985a, 1985b, 1987), and its binocular generalization (Grossberg, 1987, Grossberg & Marshall, 1989), has modeled many boundary segmentation properties of V1 and its prestriate projections. The BCS has until now been used to analyze data generated in response to static visual images. Henceforth I will therefore call such a BCS a static BCS model. Nonetheless its model cells can be gated by cells sensitive to image transients to generate receptive fields sensitive to visual motion. How does a motion BCS differ from a static BCS whose cells are sensitive to image transients?

## 2   STATIC AND MOTION FILTERING: DIRECTION-OF-CONTRAST AND DIRECTION-OF-MOTION

That boundaries of opposite direction-of-contrast are perceptually linked is vividly illustrated by the reverse-contrast Kanizsa square. A fundamental property of the front end of the BCS, which is a Static Oriented Contrast Filter (SOC Filter), is that its output is insensitive to direction-of-contrast, in order to support perception of boundaries in variable illumination. This insensitivity is achieved through the pooling by units identified with complex cells of information of units identified with simple cells, whose receptive fields are elongated and sensitive to opposite contrast polarities. The pooling implies that the complex cell layer of the SOC Filter is insensitive to direction-of-motion, as well as to direction-of-contrast. Evidently, any useful filter that will act as the front-end of a motion segmentation system must be sensitive to direction-of-motion while being insensitive to direction-of-contrast.

## 3   GLOBAL SEGMENTATION AND GROUPING: FROM LOCALLY AMBIGUOUS MOTION SIGNALS TO COHERENT OBJECT MOTION SIGNALS

In their discussion of "velocity space," Adelson and Movshon (1980, 1982) introduce diagrams similar to Figure 1a to illustrate local motion direction (and speed) ambiguity from information confined to an aperture. In Figure 1a the length of arrows codes possible trajectories of a point which would be consistent with the measured change of contrast over time of the cell in question; for this reason, it is sometimes said that early cells are sensitive to only the normal component of velocity. Figure 1B shows another view of this situation; the length of arrows is roughly proportional to a cell's "prior probability distribution" for interpreting changing stimulation as occurring in one of several directions, of which the direction perpendicular to the cell receptive field's axis of orientation is locally preferred. Note that in this conception, if a cell with an oriented receptive field (e.g. a simple cell) is being stimulated by an edge that is not perfectly aligned with its receptive field's dark-to-light contrast axis, its "preferred direction" will not correspond to that perpendicular to the edge. In this case, however, it is assumed that within a hypercolumn of cells tuned to similar spatial frequency, contrast, and temporal parameters but varying in preferred orientation, some other cell whose preferred orientation was more nearly aligned with the edge would generate a stronger signal than the cell in question. Thus, the distribution of motion signals across cells tuned to all orientations would

favor the direction perpendicular to the orientation of the edge.

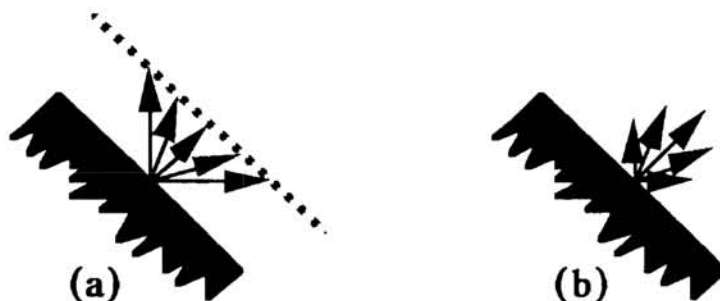

Figure 1: Motion direction ambiguity along an edge

# 4    STATIC AND MOTION COOPERATIVE GROUPING

The static BCS contains a process of for long-range completion, regularization, and grouping which is mediated by a cooperative-competitive feedback loop (CC Loop) whose competitive layer is identified with hypercomplex cells of V2 and whose cooperative layer contains units called "bipole cells," which are hypothesized to exist in the projections of V2 cells. The CC Loop seeks to form and sharpen boundaries whenever evidence from bottom-up inputs in two regions indicates that a collinear (possibly curved) continuation of boundary activity is called for. A horizontally tuned bipole cell sends feedback to horizontally tuned cells in the competitive layer.

In considering how the static CC Loop must be modified to deal with motion segmentation, consider that motion is not binary but continuously valued; headings can be, for example, "north by northwest." The analysis of moving contours thus requires one more degree of freedom than the analysis of static contours, for a contour of a given orientation can be moving in an infinity of directions, and conversely contours of any orientation can be moving in the same direction; thus a modification in the structure of the static BCS is required. Consider again the aperture problem. In the barberpole illusion the perception of motion direction along entire contours – whose measurement by cells with localized receptive fields is everywhere subject to the aperture problem – is determined by the perceived motion of their endpoints (Wallach,1976). Endstopping in simple cells of the MOC Filter can provide the enhancement of signals from segment endpoints, enabling the cooperative bipole cells of the motion CC Loop to reorganize the ambiguous local motion signals from the interiors of the diagonal segments into signals that are consistent with those of the endpoints.

# 5    GENERALIZING THE GROSSBERG-RUDD MOC FILTER FOR SEGMENTATION AND GROUPING

The original Grossberg & Rudd MOC Filter is illustrated in Figure 2. The goal is to generalize certain of its functions to handle 2-D (two-dimensional) motion segmentation issues. The MOC Filter is insensitive to direction-of-contrast but sensitive

to direction-of-motion. Level 1 registers the input pattern. Level 2 consists of sustained response cells with oriented receptive fields that are sensitive to direction-of-contrast. Level 3 consists of transient response cells with unoriented receptive fields that are sensitive to direction of change in the total cell input. Level 4 cells combine sustained cell and transient cell signals to become sensitive to direction-of-motion and sensitive to direction-of-contrast. Level 5 cells combine Level 4 cells to become sensitive to direction-of-motion and insensitive to direction-of-contrast.

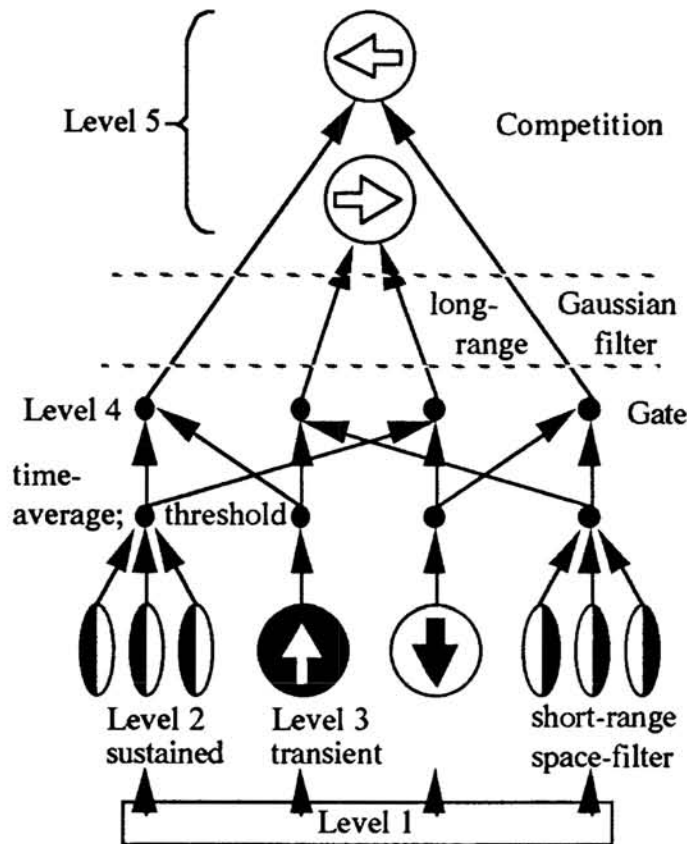

Figure 2: The Motion Oriented Contrast (MOC) Filter

The full domain of motion segmentation and grouping includes such problems as determining structure in depth from motion, motion transparency, and motion grouping amid occlusion. Although the motion BCS is conceived with these and related difficult phenomena in mind, I will instead focus on the elementary grouping operations necessary to perform detections of object motion within the visual field. Even here difficult issues arise. Consider the lower right corner of a homogeneous rectangular form of relatively high luminance that is moving diagonally upward and to the right on a homogeneous background of relatively low luminance. (See Figure 3a.) In region A dark-to-light (luminance increasing over time) transition occurs at a vertical edge, while in region B a light-to-dark (luminance decreasing over time) transition occurs at a horizontal edge. Both the regions of horizontal and vertical contrast near the corner provide signals to the MOC Filter, provided that the sustained cells of Level 2 (Figure 2) are taken to be spatially laid out as indicated in Figure 3b. Over three successive time increments, the contours of the

rectangle of Figure 3a occur in the positions indicated, while luminance increases along the vertical edge and decreases along the horizontal edge. If certain of the sustained cell receptive fields sending inputs to Level 4 of the MOC Filter (Figure 2) were arranged as indicated, a diagonal motion signal could be generated from both vertically and horizontally oriented cells, in conjunction with luminance gating signals of opposite signs. (Of course, motion signals of many other directions will also be generated along the lengths of the horizontal and vertical edges; these will be considered subsequently.) In other words, for at least some of the gating nodes of Layer 4 (Figure 2), the layout of receptive field centers of contributing sustained cells of Layer 2 is taken to be in a direction diagonal to the orientational preference of the individual sustained cells. It would make no sense to build a motion filter whose receptive field centers were arrayed collinearly with the contributing sustained cell's orientational preference – although this type of arrangement might be suitable for collinear completion in a static form system. Accordingly, it appears that a variant of a "sine law" exists, whereby the contribution of any sustained cell at Level 2 to Level 4 gating cell is modulated by the (absolute value of the) sine of the angle formed between the sustained cell's orientational preference and the gating cell's directional preference.

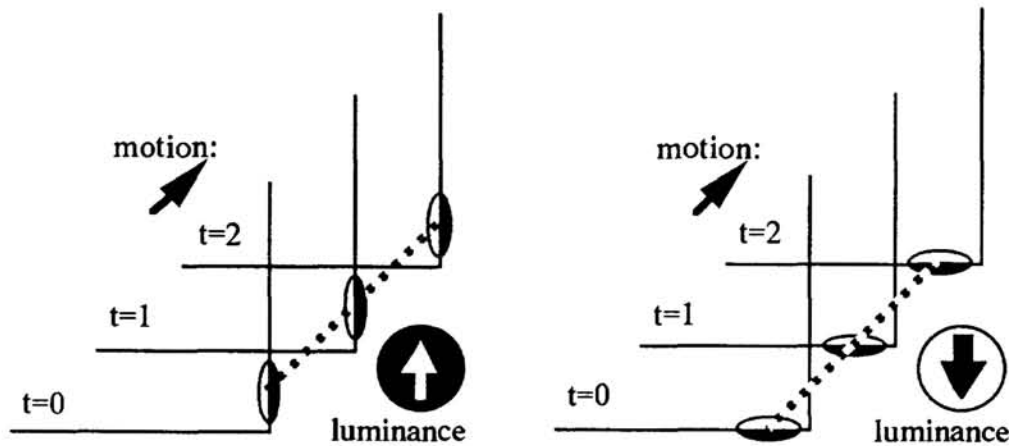

Figure 3: The corner of a light rectangle moving diagonally

The long range filter (Level 5, Figure 2) can simultaneously accept motion signals from both the horizontal and vertical edges of the moving corner, despite the gating of one set of signals by transient "luminance increasing" detectors (Level 3, Figure 2) and gating of another set by "luminance decreasing" detectors. Thus while simultaneous increase and decrease of luminance is logically impossible in an infinitesimal area, and while a too rapid change from increase to decrease may be unresolvable by sustained cells at Level 2, the simultaneous nearby increase and decrease of luminance with a coherent trajectory or direction despite different contour orientations is fodder for the long-range filter. Note that the long-range filter of the MOC Filter is not the same as the long-range grouping stage of the CC Loop.

# 6  ENDSTOPPING: GENERATION OF A TERMINATOR OR CORNER ADVANTAGE IN MOTION SIGNALS

In discussing the barberpole illusion I referred to an "advantage" for motion signals near terminators or corners of contours. The designation "advantage" connotes that those signals tend to be better indicators of object motion than signals generated from a relatively straight interior of a contour. For this advantage to be manifest in perception, however, that advantage must also be one of signal strength, the more so because the regions or spatial extent of interior motion signals is often larger than the region of terminator or corner signals. The source of the advantage would appear to involve endstopping at the very front end of the MOC Filter. Many simple cells, identified with the orientation and direction-of-contrast sensitive sustained cells of Level 2 of the MOC Filter, exhibit endstopping (Dreher, 1972.) (Note that this endstopping is functionally analogous to the first competitive stage of the SOC Filter.) Strong endstopping, whereby only signals at terminators survive, can reduce the problem of determining motion direction to one of tracking an isolated region of activity. In the case of weak endstopping considered here, however, surviving signals indicating "locally preferred" directions can continue to confound the problem of motion segmentation and grouping.

# 7  CONSENSUS AT CORNERS: GAUSSIAN SPACE AVERAGING AND DIRECTIONAL COMPETITION

In the weak endstopping case the local motion signals from the lower right corner of the moving rectangle would have roughly the form diagramed in Figure 4a. While there is some preference for diagonal (up-and-to-the-right) signals, local motion signals of other directions also exist. (b) A mechanism is needed to combine different directions signals into a single coherent local direction signal. (c) The signal combination can be accomplished by a motion analog of the second competitive stage among orientations of the SOC Filter, as described in Grossberg & Mingolla, (1987). A excitatory on-center, inhibitory off-surround network organization among cells coding different directions-of-motion at the same position can accomplish the desired pooling and choice through competitive peak summation and sharpening. Note that the domain of spatial averaging of the Gaussian filter (transition from Level 4 to Level 5 of the MOC Filter) is presumed to be large enough to span the signals generated by the ends of the leading vertical and trailing horizontal edges. At Level 5, then, signals of many directions occur for cells coding the same position. Those directions will have the appropriate "central tendency", however, and a simple center-surround competition in the space of directions, analogous to the revised version of the second competitive stage (for orientations) of the static BCS described by Grossberg & Mingolla (1987), suffices to choose the direction which is most consistent with surrounding input data at each location. (See Figure 4b.)

In this article I have described motion analysis mechanisms whereby the visual system frees itself from an excessive reliance on either purely local (short-range filtering) computations or top-down (cognitive or expectancy based) computations. Instead, within a perceptual middle ground, competitive and cooperative interac-

tions withing a parallel and structured network with several scales of interaction help to choose and enhance those aspects of local data which contribute to coherent and consistent measures of object motion.

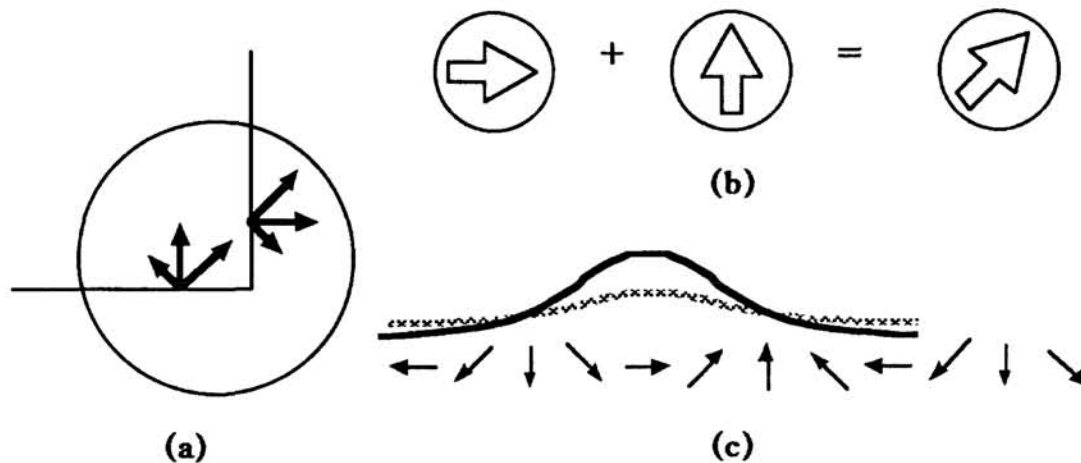

Figure 4: Resolution of ambiguous signals at corners

## Acknowledgements

The research described was performed jointly with Stephen Grossberg.

The author was supported in part by AFOSR F49620-87-C-0018.

## References

Adelson, E. H. & Movshon, J. A. (1980). *Journal of the Optical Society of America,* **70**, 1605.

Adelson, T. & Movshon, J. A. (1982). *Nature,* **300**, 523-525.

Dreher, B. (1972) investigative Ophthamology, **11**, 355-356.

Grossberg, S. (1987) *Perception and Psychophysics,* **41**, 87-116.

Grossberg, S. & Marshall, J. (1989). *Neural Networks,* **2**, 29-51.

Grossberg, S. & Mingolla, E. (1985a). *Psychological Review,* **92**, 173-211.

Grossberg, S. & Mingolla, E. (1985b). *Perception and Psychophysics,* **38**, 141-171.

Grossberg, S. & Mingolla, E. (1987). *Computer Vision, Graphics, and Image Processing,* **37**, 116-165.

Grossberg, S. & Rudd, M. (1989). *Neural Networks,* **2**, 421-450.

Wallach, H. (1976). *On perception.* New York, Quadrangle.